# Deep Learning with Kernel Regularization for Visual Recognition

**Kai Yu      Wei Xu      Yihong Gong**
NEC Laboratories America, Cupertino, CA 95014, USA
{kyu, wx, ygong}@sv.nec-labs.com

## Abstract

In this paper we aim to train deep neural networks for rapid visual recognition. The task is highly challenging, largely due to the lack of a meaningful regularizer on the functions realized by the networks. We propose a novel regularization method that takes advantage of kernel methods, where an oracle kernel function represents prior knowledge about the recognition task of interest. We derive an efficient algorithm using stochastic gradient descent, and demonstrate encouraging results on a wide range of recognition tasks, in terms of both accuracy and speed.

## 1  Introduction

Visual recognition remains a challenging task for machines. This difficulty stems from the large pattern variations under which a recognition system must operate. The task is extremely easy for a human, largely due to the expressive deep architecture employed by human visual cortex systems. Deep neural networks (DNNs) are argued to have a greater capacity to recognize a larger variety of visual patterns than shallow models, because they are considered biologically plausible.

However, training deep architectures is difficult because the large number of parameters to be tuned necessitates an enormous amount of labeled training data that is often unavailable. Several authors have recently proposed training methods by using unlabeled data. These methods perform a greedy layer-wise pre-training using unlabeled data, followed by a supervised fine-tuning [9, 4, 15]. Even though the strategy notably improves the performance, to date, the best reported recognition accuracy on popular benchmarks such as Caltech101 by deep models is still largely behind the results of shallow models.

Beside using unlabeled data, in this paper we tackle the problem by leveraging additional *prior knowledge*. In the last few decades, researchers have developed successful kernel-based systems for a wide range of visual recognition tasks. Those sensibly-designed kernel functions provide an extremely valuable source of prior knowledge, which we believe should be exploited in deep learning. In this paper, we propose an *informative* kernel-based regularizer, which makes it possible to train DNNs with prior knowledge about the recognition task.

Computationally, we propose to solve the learning problem using *stochastic gradient descent* (SGD), as it is the *de facto* method for neural network training. To this end we transform the kernel regularizer into a loss function represented as a sum of costs by individual examples. This results in a simple multi-task architecture where a number of extra nodes at the output layer are added to fit a set of auxiliary functions automatically constructed from the kernel function.

We apply the described method to train convolutional neural networks (CNNs) for a wide range of visual recognition tasks, including handwritten digit recognition, gender classification, ethnic origin recognition, and object recognition. Overall our approach exhibits excellent accuracy and speed on all of these tasks. Our results show that incorporation of prior knowledge can boost the performance of CNNs by a large margin when the training set is small or the learning problem is difficult.

## 2  DNNs with Kernel Regularization

In our setting, the learning model, a deep neural network (DNN), aims to learn a predictive function $f : \mathcal{X} \to \mathbb{R}$ that can achieve a low expected discrepancy $E[\ell(y, f(x))]$ over the distribution $p(x, y)$. In the simplest case $\mathcal{Y} = \{-1, 1\}$ and $\ell(\cdot, \cdot)$ is a differentiable hinge loss. Based on a set of labeled examples $[(x_i, y_i)]_{i=1}^n$, the learning is by minimizing a regularized loss

$$L(\beta, \theta) = \sum_{i=1}^n \ell\left(y_i, \beta_1^\top \phi_i + \beta_0\right) + \lambda \|\beta_1\|^2 \tag{1}$$

where $\phi_i = \phi(x_i; \theta)$ maps $x_i$ to $q$-dimensional hidden units via a nonlinear deep architecture with parameters $\theta$, including the connection weights and biases of all the intermediate layers, $\beta = \{\beta_1, \beta_0\}$, $\beta_1$ includes all the parameters of the transformation from the last hidden layer to the output layer, $\beta_0$ is a bias term, $\lambda > 0$, and $\|a\|^2 = \text{tr}(a^\top a)$ is the usual weight decay regularization. Applying the well-known representor theorem, we derive the equivalence to a kernel system[1]

$$L(\alpha, \beta_0, \theta) = \sum_{i=1}^n \ell\left(y_i, \sum_{j=1}^n \alpha_j K_{i,j} + \beta_0\right) + \lambda \sum_{i,j=1}^n \alpha_i \alpha_j K_{i,j} \tag{2}$$

where the kernel is computed by

$$K_{i,j} = \langle \phi(x_i; \theta), \phi(x_j; \theta) \rangle = \phi_i^\top \phi_j$$

We assume the network is provided with some prior knowledge, in the form of an $m \times m$ kernel matrix $\Sigma$, computed on $n$ labeled training data, plus possibly additional $m - n$ *unlabeled data* if $m > n$. We exploit this prior knowledge via imposing a kernel regularization on $K(\theta) = [K_{i,j}]_{i,j=1}^m$, such that the learning problem seeks

**Problem 2.1.**

$$\min_{\beta, \theta} \quad L(\beta, \theta) + \gamma \Omega(\theta) \tag{3}$$

*where $\gamma > 0$ and $\Omega(\theta)$ is defined by*

$$\Omega(\theta) = \text{tr}\left[K(\theta)^{-1} \Sigma\right] + \log \det[K(\theta)] \tag{4}$$

This is a case of *semi-supervised learning* if $m > n$. Though $\Omega$ is non-convex w.r.t. $K$, it has a unique minimum at $K = \Sigma$ if $\Sigma \succ 0$, suggesting that minimizing $\Omega(\theta)$ encourages $K$ to approach $\Sigma$. The regularization can be explained from an information-theoretic perspective. Let $p(f|K)$ and $p(f|\Sigma)$ be two Gaussian distributions $\mathcal{N}(0, K)$ and $\mathcal{N}(0, \Sigma)$.[2] $\Omega(\theta)$ is related to the KL-divergence $D_{KL}[p(f|\Sigma)\|p(f|K)]$. Therefore, minimizing $\Omega(\theta)$ forces the two distributions to be close. We note that the regularization does not require $\Sigma$ to be positive definite — it can be semidefinite.

## 3  Kernel Regularization via Stochastic Gradient Descent

The learning problem in Eq. (3) can be solved by using gradient-based methods. In this paper we emphasize large-scale optimizations using stochastic gradient descent (SGD), because the method is fast when the size $m$ of total data is large and backpropagation, a typical SGD, has been the *de facto* method to train neural networks for large-scale learning tasks.

SGD considers the problem where the optimization cost is the sum of the local cost of each individual training example. A standard batch gradient descent updates the model parameters by using the true gradient summed over the whole training set, while SGD approximates the true gradient by the gradient caused by a single random training example. Therefore, the parameters of the model

are updated after each training example. For large data sets, SGD is often much faster than batch gradient descent.

However, because the regularization term defined by Eq. (4) does not consist of a cost function that can be expressed as a sum (or an average) over data examples, SGD is not directly applicable. Our idea is to transform the problem into an equivalent formulation that can be optimized stochastically.

## 3.1 Shrinkage on the Kernel Matrix

We consider a large-scale problem where the data size $m$ may grow over time, while the size of the last hidden layer ($q$) of the DNN is fixed. Therefore the computed kernel $K$ can be rank deficient. In order to ensure that the trace term in $\Omega(\theta)$ is well-defined, and that the log-determinant term is bounded from below, we instead use $K + \delta I$ to replace $K$ in $\Omega(\theta)$, where $\delta > 0$ is a small shrinkage parameter and $I$ is an identity matrix. Thus the log-determinant acts on a much smaller $q \times q$ matrix[3]

$$\log \det(K + \delta I) = \log \det \left( \Phi^\top \Phi + \delta I \right) + const$$

where $\Phi = [\phi_1, \ldots, \phi_m]^\top$ and $const = (m - q) \cdot \log \delta$. Omitting all the irrelevant constants, we then turn the kernel regularization into

$$\Omega(\theta) = \mathrm{tr}\left[ (\Phi\Phi^\top + \delta I)^{-1}\Sigma \right] + \log \det(\Phi^\top \Phi + \delta I) \tag{5}$$

The kernel shrinkage not only remedies the ill-posedness, but also yields other conveniences in our later development.

## 3.2 Transformation of the Log-determinant Term

By noticing that $\Phi^\top \Phi = \sum_{i=1}^n \phi_i \phi_i^\top$ is a sum of quantities over data examples, we move it outside of the log determinant for the convenience of SGD.

**Theorem 3.1.** *Consider* $\min_\theta \{L(\theta) = h(\theta) + g(a)\}$, *where* $g(\cdot)$ *is concave and* $a \equiv a(\theta)$ *is a function of* $\theta$, *if its local minimum w.r.t.* $\theta$ *exists, then the problem is equivalent to*

$$\min_{\theta, \psi} \left\{ L(\theta, \psi) = h(\theta) + a(\theta)^\top \psi - g^\bullet(\psi) \right\} \tag{6}$$

*where* $g^\bullet(\psi)$ *is the conjugate function of* $g(a)$, *i.e.* $g^\bullet(\psi) = \min_a \{\psi^\top a - g(a)\}$.[4]

*Proof.* For a concave function $g(a)$, the conjugate function of its conjugate function is itself, i.e., $g(a) = \min_\psi \{a^\top \psi - g^\bullet(\psi)\}$. Since $g^\bullet(\psi)$ is concave, $a^\top \psi - g^\bullet(\psi)$ is convex w.r.t. $\psi$ and has the unique minimum $g(a)$. Therefore minimizing $L(\theta, \psi)$ w.r.t. $\theta$ and $\psi$ is equivalent to minimizing $L(\theta)$ w.r.t. $\theta$. □

Since log-determinant is concave for $q \times q$ positive definite matrices $A$, the conjugate function of $\log \det(A)$ is $\log \det(\Psi) + q$. We can use the above theorem to transform any loss function containing $\log \det(A)$ into another loss, which is an upper bound and involves $A$ in a linear term. Therefore the log-determinant in Eq. (5) is turned into a variational representation

$$\log \det \left( \Phi^\top \Phi + \delta I \right) = \min_{\Psi \in \mathbb{S}_q^+} \left[ \sum_{i=1}^m \phi_i^\top \Psi \phi_i + \delta \cdot \mathrm{tr}(\Psi) - \log \det(\Psi) + const \right]$$

where $\Psi \in \mathbb{S}_q^+$ is a $q \times q$ positive definite matrix, and $const = -q$. As we can see, the upper bound is a convex function of auxiliary variables $\Psi$ and more importantly, it amounts to a sum of local quantities caused by each of the $m$ data examples.

### 3.3 Transformation of the Trace Term

We assume that the kernel matrix $\Sigma$ is presented in a decomposed form $\Sigma = UU^\top$, with $U = [u_1, \ldots, u_m]^\top$, $u_i \in \mathbb{R}^p$, and $p \leq m$. We have found that the trace term can be cast as a variational problem by introducing an $q \times p$ auxiliary variable matrix $\eta$.

**Proposition 3.1.** *The trace term in Eq. (5) is equivalent to a convex variational representation*

$$\text{tr}\left[(\Phi\Phi^\top + \delta I)^{-1}\Sigma\right] = \min_{\eta \in \mathbb{R}^{q \times p}}\left[\sum_{i=1}^{m}\|\frac{1}{\sqrt{\delta}}u_i - \eta^\top\phi_i\|^2 + \delta\|\eta\|_F^2\right]$$

*Proof.* We first obtain the analytical solution $\eta^* = \frac{1}{\sqrt{\delta}}(\Phi^\top\Phi + \delta I)^{-1}\Phi^\top U$, where the variational representation reaches its unique minimum. Then, plugging it back into the function, we have

$$\text{tr}\left[\frac{1}{\delta}U^\top U - 2\frac{1}{\sqrt{\delta}}U^\top\Phi\eta^* + {\eta^*}^\top\Phi^\top\Phi\eta^* + \frac{1}{\delta}U^\top\Phi(\Phi^\top\Phi + \delta I)^{-2}\Phi^\top U\right]$$

$$= \frac{1}{\delta}\text{tr}\left[U^\top U - U^\top\Phi(\Phi^\top\Phi + \delta I)^{-1}\Phi^\top U\right] = \text{tr}\left[(\Phi\Phi^\top + \delta I)^{-1}UU^\top\right]$$

where the last step is derived by applying the Woodbury matrix identity. $\square$

Again, we note that the upper bound is a convex function of $\eta$, and consists of a sum of local costs over data examples.

### 3.4 An Equivalent Learning Framework

Combining the previous results, we obtain the convex upper bound for the kernel regularization Eq. (5), which amounts to a sum of costs over examples under some regularization

$$\Omega(\theta) \leq \left[L(\eta, \Psi, \theta) = \sum_{i=1}^{m}\left(\|\frac{1}{\sqrt{\delta}}u_i - \eta^\top\phi_i\|^2 + \phi_i^\top\Psi\phi_i\right) + \delta\|\eta\|_F^2 + \delta \cdot \text{tr}(\Psi) - \log\det(\Psi)\right]$$

where we omit all the terms irrelevant to $\eta, \Psi$ and $\theta$. $L(\eta, \Psi, \theta)$ is convex w.r.t. $\eta$ and $\Psi$, and has a unique minimum $\Omega(\theta)$, hence we can replace $\Omega(\theta)$ by instead minimizing the upper bound and formulate an equivalent learning problem

$$\min_{\beta, \eta, \Psi, \theta}\left[L(\beta, \eta, \Psi, \theta) = L(\beta, \theta) + \gamma L(\eta, \Psi, \theta)\right] \tag{7}$$

Clearly this new optimization can be solved by SGD.

When applying the SGD method, each step based on one example needs to compute the inverse of $\Psi$. This can be computationally unaffordable when the dimensionality is large (e.g. $q > 1000$) — remember that the efficiency of SGD is dependent on the lightweight of each stochastic update. Our next result suggests that we can dramatically reduce this complexity from $O(q^3)$ to $O(q)$.

**Proposition 3.2.** *Eq. (5) is equivalent to the convex variational problem*

$$\Omega(\theta) = \min_{\eta, \psi}\left[\sum_{i=1}^{m}\left(\|\frac{1}{\sqrt{\delta}}u_i - \eta^\top\phi_i\|^2 + \psi^\top\phi_i^2\right) + \delta\|\eta\|_F^2 + \delta \cdot \psi^\top e - \sum_{k=1}^{q}\log\psi_k\right] \tag{8}$$

*where $\psi = [\psi_1, \ldots, \psi_q]^\top$, and $e = [1, \ldots, 1]^\top$.*

*Proof.* There is an ambiguity of the solutions up to rotations. Suppose $\{\beta^*, \Phi^*, \eta^*, \Psi^*\}$ is an optimal solution set, a transformation $\beta^* \leftarrow R\beta^*$, $\Phi^* \leftarrow R\Phi^*$, $\eta^* \leftarrow R\eta^*$, and $\Psi^* \leftarrow R\Psi^*R^\top$ results in the same optimality if $R^\top R = I$. Since there always exists an $R$ to diagonalize $\Psi^*$, we can pre-restrict $\Psi$ to be a diagonal positive definite matrix $\Psi = \text{diag}[\psi_1, \ldots, \psi_q]$, which does not change our problem and gives rise to Eq. (8). $\square$

We note that the variational form is convex w.r.t. the auxiliary variables $\eta$ and $\psi$. Therefore we can formulate the whole learning problem as

**Problem 3.1.**

$$\min_{\beta,\eta,\psi,\theta} \left[ L(\beta,\eta,\psi,\theta) = \frac{1}{n}L_1(\beta,\theta) + \frac{\gamma}{mn}L_2(\eta,\theta) + \frac{\gamma}{mn}L_3(\psi,\theta) \right] \tag{9}$$

*where $L_1(\beta,\theta)$ is defined by Eq. (1), and*

$$L_2(\eta,\theta) = \sum_{i=1}^{m} \|\frac{1}{\sqrt{\delta}}u_i - \eta^\top \phi_i\|^2 + \delta\|\eta\|_F^2$$

$$L_3(\psi,\theta) = \sum_{i=1}^{m} \psi^\top \phi_i^2 + \delta \cdot \psi^\top e - \sum_{k=1}^{q} \log \psi_k$$

To ensure the estimator of $\beta$ and $\theta$ is consistent, the effect of regularization should vanish as $n \to \infty$. Therefore we intentionally normalize $L_2(\eta,\theta)$ and $L_3(\psi,\theta)$ by $1/m$. The overall loss function is averaged over the $n$ labeled examples, consisting of *three loss functions*: the main classification task $L_1(\beta,\theta)$, an auxiliary least-squares regression problem $L_2(\eta,\theta)$, and an additional regularization term $L_3(\psi,\theta)$, which can be interpreted as another least-squares problem. Since each of the loss functions amounts to a summation of local costs caused by individual data examples, the whole learning problem can be conveniently implemented by SGD, as described in Algorithm 1.

In practice, the kernel matrix $\Sigma = UU^\top$ that represents domain knowledge can be obtained in three different ways: (i) In the easiest case, $U$ is directly available by computing some hand-crafted features computed from the input data, which corresponds to a case of a linear kernel function; (ii) $U$ can be results of some unsupervised learning (e.g. the self-taught learning [14] based on sparse coding), applied on a large set of unlabeled data; (iii) If a nonlinear kernel function is available, $U$ can be obtained by applying incomplete Cholesky decomposition on an $m \times m$ kernel matrix $\Sigma$. In the third case, when $m$ is so large that the matrix decomposition cannot be computed in the main memory, we apply the Nyström method [19]: We first randomly sample $m_1$ examples $p < m_1 < m$, such that the computed kernel matrix $\Sigma_1$ can be decomposed in the memory. Let $VDV^\top$ be the $p$-rank eigenvalue decomposition of $\Sigma_1$, then the $p$-rank decomposition of $\Sigma$ can be approximated by $\Sigma \approx UU^\top, U = \Sigma_{:,1}VD^{-\frac{1}{2}}$, where $\Sigma_{:,1}$ is the $m \times m_1$ kernel matrix between all the $m$ examples and the subset of size $m_1$.

---

**Algorithm 1** Stochastic Gradient Descent

---

**repeat**
    Generate a number $a$ from uniform distribution $[0,1]$
    **if** $a < \frac{n}{m+n}$ **then**
        Randomly pick a sample $i \in \{1, \cdots, n\}$ for $L_1$, and update parameter by

$$[\beta,\theta] \leftarrow [\beta,\theta] - \epsilon \frac{\partial L_1(x_i,\beta,\theta)}{\partial [\beta,\theta]}$$

    **else**
        Randomly pick a sample $i \in \{1, \cdots, m\}$ for $L_2$, and update parameter by

$$[\eta,\psi,\theta] \leftarrow [\eta,\psi,\theta] - \frac{\epsilon}{m} \frac{\partial[L_2(x_i,\eta,\theta) + L_3(x_i,\psi,\theta)]}{\partial[\eta,\psi,\theta]}$$

    **end if**
**until** convergence

---

## 4   Visual Recognition by Deep Learning with Kernel Regularization

In the following, we apply the proposed strategy to train a class of deep models and convolutional neural networks (CNNs, [11]) for a range of visual recognition tasks including digit recognition on MNIST dataset, gender and ethnicity classification on the FRGC face dataset, and object recognition on the Caltech101 dataset. In each of these tasks, we choose a kernel function that has been reported to have state-of-the-art or otherwise good performances in the literature. We will see whether a kernel-regularizer can improve the recognition accuracy of the deep models, and how it is compared with the support vector machine (SVM) using the exactly the same kernel.

Table 1: Percentage error rates of handwritten digit recognition on MNIST

| Training Size | 100 | 600 | 1000 | 3000 | 60000 |
|---|---|---|---|---|---|
| SVM (RBF) | 22.73 | 8.53 | 6.58 | 3.91 | 1.41 |
| SVM (RBF, Nyström) | 24.73 | 9.15 | 6.92 | 5.51 | 5.16 |
| SVM (Graph) | 5.21 | 3.74 | 3.46 | 3.01 | 2.23 |
| SVM (Graph, Cholesky) | 7.17 | 6.47 | 5.75 | 4.28 | 2.87 |
| CNN | 19.40 | 6.40 | 5.50 | 2.75 | 0.82 |
| kCNN (RBF) | 14.49 | 3.85 | 3.40 | 1.88 | 0.73 |
| kCNN (Graph) | 4.28 | 2.36 | 2.05 | 1.75 | 0.64 |
| CNN (Pretrain) [15] | – | 3.21 | – | – | 0.64 |
| Embed$^O$ CNN [18] | 11.73 | 3.42 | 3.34 | 2.28 | – |
| Embed$^{I5}$ CNN [18] | 7.75 | 3.82 | 2.73 | 1.83 | – |
| Embed$^{A1}$ CNN [18] | 7.87 | 3.82 | 2.76 | 2.07 | – |

Throughout all the experiments, "kCNN" denotes CNNs regularized by nonlinear kernels, processed by either Cholesky or Nyström approximation, with parameters $p = 600$, $m_1 = 5000$, and $m$ the size of each whole data set. The obtained $u_i$ are normalized to have unitary lengths. $\lambda$ and $\delta$ are fixed by 1. The remaining two hyperparameters are: the learning rates $\epsilon = \{10^{-3}, 10^{-4}, 10^{-5}\}$ and the kernel regularization weights $\gamma = \{10^2, 10^3, 10^4, 10^5\}$. Their values are set once for each of the 4 recognition tasks based on a 5-fold cross validation using 500 labeled examples.

## 4.1 Handwritten Digit Recognition on MNIST Dataset

The data contains a training set with 60000 examples and a test set with 10000 examples. The CNN employs 50 filters of size $7 \times 7$ on $34 \times 34$ input images, followed by down-sampling by $1/2$, then 128 filters of size $5 \times 5$, followed by down-sampling by $1/2$, and then 200 filters of size $5 \times 5$, giving rise to 200 dimensional features that are fed to the output layer. Two nonlinear kernels are used: (1) RBF kernel, and (2) Graph kernel on 10 nearest neighbor graph [6]. We perform 600-dimension Cholesky decomposition on the whole $70000 \times 70000$ graph kernel because it is very sparse.

In addition to using the whole training set, we train the models on $100, 600, 1000$ and $3000$ random examples from the training set and evaluate the classifiers on the whole test set, and repeat each setting by 5 times independently. The results are given in Tab. 1. kCNNs effectively improve over CNNs by leveraging the prior knowledge, and also outperform SVMs that use the same kernels. The results are competitive with the state-of-the-art results by [15], and [18] of a different architecture.

## 4.2 Gender and Ethnicity Recognition on FRGC Dataset

The FRGC 2.0 dataset [13] contains 568 individuals' 14714 face images under various lighting conditions and backgrounds. Beside person identities, each image is annotated with gender and ethnicity, which we put into 3 classes, "white", "asian", and "other". We fix 114 persons' 3014 images (randomly chosen) as the testing set, and randomly selected $5\%, 10\%, 20\%, 50\%$, and "All" images from the rest 454 individuals' 11700 images. For each training size, we randomize the training data 5 times and report the average error rates.

In this experiment, CNNs operate on images represented by R/G/B planes plus horizontal and vertical gradient maps of gray intensities. The 5 input planes of size $140 \times 140$ are processed by 16 convolution filters with size $16 \times 16$, followed by max pooling within each disjoint $5 \times 5$ neighborhood. The obtained 16 feature maps of size $25 \times 25$ are connected to the next layer by 256 filters of size $6 \times 6$, with $50\%$ random sparse connections, followed by max pooling within each $5 \times 5$ neighborhood. The resulting $256 \times 4 \times 4$ features are fed to the output layer. The nonlinear kernel used in this experiment is the RBF kernel computed directly on images, which has demonstrated state-of-the-art accuracy for gender recognition [3]. The results shown in Tab. 2 and Tab. 3 demonstrate that kCNNs significantly boost the recognition accuracy of CNNs for both gender and ethnicity recognition. The difference is prominent when small training sets are presented.

## 4.3 Object Recognition on Caltech101 Dataset

Caltech101 [7] contains 9144 images from 101 object categories and a background category. It is considered one of the most diverse object databases available today, and is probably the most popular benchmark for object recognition. We follow the common setting to train on 15 and 30 images per class and test on the rest. Following [10], we limit the number of test images to 30 per class. The

Table 2: Percentage error rates of gender recognition on FRGC

| Training Size | 5% | 10% | 20% | 50% | All |
|---|---|---|---|---|---|
| SVM (RBF) | 16.7 | 13.4 | 11.3 | 9.1 | 8.6 |
| SVM (RBF, Nyström) | 20.2 | 14.3 | 11.6 | 9.1 | 8.8 |
| CNN | 61.5 | 17.2 | 8.4 | 6.6 | 5.9 |
| kCNN | 17.1 | 7.2 | 5.8 | 5.0 | 4.4 |

Table 3: Percentage error rates of ethnicity recognition on FRGC

| Training Size | 5% | 10% | 20% | 50% | All |
|---|---|---|---|---|---|
| SVM (RBF) | 22.9 | 16.9 | 14.1 | 11.3 | 10.2 |
| SVM (RBF, Nyström) | 24.7 | 20.6 | 15.8 | 11.9 | 11.1 |
| CNN | 30.0 | 13.9 | 10.0 | 8.2 | 6.3 |
| kCNN | 15.6 | 8.7 | 7.3 | 6.2 | 5.8 |

recognition accuracy was normalized by class sizes and evaluated over 5 random data splits. The CNN has the same architecture as the one used in the FRGC experiment. The nonlinear kernel is the spatial pyramid matching (SPM) kernel developed in [10].

Tab. 4 shows our results together with those reported in [12, 15] using deep hierarchical architectures. The task is much more challenging than the previous three tasks for CNNs, because in each category the data size is very small while the visual patterns are highly diverse. Thanks to the regularization by SPM kernel, kCNN dramatically improves the accuracy of CNN, and outperforms SVM using the same kernel. This is perhaps the best performance by (trainable and hand-crafted) deep hierarchical models on the Caltech101 dataset. Some filters trained with and without kernel regularization are visualized in Fig. 1, which helps to understand the difference made by kCNN.

## 5 Related Work, Discussion, and Conclusion

Recent work on deep visual recognition models includes [17, 12, 15]. In [17] and [12] the first layer consisted of hard-wired Gabor filters, and then a large number of patches were sampled from the second layer and used as the basis of the representation which was then used to train a discriminative classifier.

Deep models are powerful in representing complex functions but very difficult to train. Hinton and his coworkers proposed training deep belief networks with layer-wise unsupervised pre-training, followed by supervised fine-tuning [9]. The strategy was subsequently studied for other deep models like CNNs [15], autoassociators [4], and for document coding [16]. In recent work [18], the authors proposed training a deep model jointly with an unsupervised embedding task, which led to improved results as well. Though using unlabeled data too, our work differs from previous work at the emphasis on leveraging the prior knowledge, which suggests that it can be combined with those approaches, including neighborhood component analysis [8], to further enhance the deep learning. This work is also related to transfer learning [2] that used auxiliary learning tasks to learn a linear feature mapping, and more directly, our previous work [1], which created pseudo auxiliary tasks based on hand-craft image features to train nonlinear deep networks.

One may ask, why bother training with kCNN, instead of simply combining two independently trained CNN and SVM systems? The reason is computational speed – kCNN pays an extra cost to exploit a kernel matrix in the training phase, but in the prediction phase the system uses CNN alone.

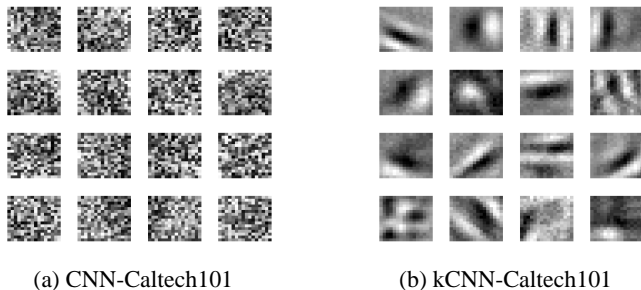

(a) CNN-Caltech101          (b) kCNN-Caltech101

Figure 1: First-layer filters on the B channel, learned from Caltech101 (30 examples per class)

Table 4: Percentage accuracy on Caltech101

| Training Size | 15 | 30 | Training Size | 15 | 30 |
|---|---|---|---|---|---|
| SVM (SPM) [10] | 54.0 | 64.6 | CNN (Pretrain) [15] | – | 54.0 |
| SVM (SPM, Nyström) | 52.1 | 63.1 | CNN | 26.5 | 43.6 |
| HMAX [12] | 51.0 | 56.0 | kCNN | 59.2 | 67.4 |

In our Caltech101 experiment, the SVM (SPM) needed several seconds to process a new image on a PC with a 3.0 GHz processor, while kCNN can process about 40 images per second. The latest record on Caltech101 was based on combining multiple kernels [5]. We conjecture that kCNN could be further improved by using multiple kernels without sacrificing recognition speed.

To conclude, we proposed using kernels to improve the training of deep models. The approach was implemented by stochastic gradient descent, and demonstrated excellent performances on a range of visual recognition tasks. Our experiments showed that prior knowledge could significantly improve the performance of deep models when insufficient labeled data were available in hard recognition problems. The trained model was much faster than kernel systems for making predictions.

**Acknowledgment**: We thank the reviewers and Douglas Gray for helpful comments.

## Footnotes

[1]In this paper we slightly abuse the notation, i.e., we use $L$ to denote different loss functions. However their meanings should be uniquely identified by checking the input parameters.

[2]From a Gaussian process point of view, a kernel function defines the prior distribution of a function $f$, such that the marginal distribution of the function values $f$ on any finite set of inputs is a multivariate Gaussian.

[3]Hereafter in this paper, with a slight abuse of notation, we use "$const$" in equations to summarize the terms irrelevant to the variables of interest.

[4]If $g(a)$ is convex, its conjugate function is $g^\circ(\psi) = \max_a \{\psi^\top a - g(a)\}$.

# References

[1] A. Ahmed, K. Yu, W. Xu, Y. Gong, and E. P. Xing. Training hierarchical feed-forward visual recognition models using transfer learning from pseudo tasks. *European Conference on Computer Vision*, 2008.

[2] R. K. Ando and T. Zhang. A framework for learning predictive structures from multiple tasks and unlabeled data. *Journal of Machine Learning Research*, 2005.

[3] S. Baluja and H. Rowley. Boosting sex identification performance. *Journal of Computer Vision*, 2007.

[4] Y. Bengio, P. Lamblin, D. Popovici, and H. Larochelle. Greedy layer-wise training of deep networks. *Neural Information Processing Systems*, 2007.

[5] A. Bosch, A. Zisserman, and X. Munõz. Image classification using ROIs and multiple kernel learning. 2008. submitted to International Journal of Computer Vison.

[6] O. Chapelle, J. Weston, and B. Schölkopf. Cluster kernels for semi-supervised learning. *Neural Information Processing Systems*, 2003.

[7] L. Fei-Fei, R. Fergus, and P. Perona. Learning generative visual models from few training examples: An incremental Bayesian approach tested on 101 object categories. *CVPR Workshop*, 2004.

[8] J. Goldberger, S. Roweis, G. Hinton, and R. Salakhutdinov. Neighbourhood components analysis. *Neural Information Processing Systems*, 2005.

[9] G. E. Hinton and R. R. Salakhutdinov. Reducing the dimensionality of data with neural networks. *Science*, 313(5786):504 – 507, July 2006.

[10] S. Lazebnik, C. Schmid, and J. Ponce. Beyond bags of features: Spatial pyramid matching for recognizing natural scene categories. *IEEE Conference on Computer Vision and Pattern Recognition*, 2006.

[11] Y. LeCun, L. Bottou, Y. Bengio, and P. Haffner. Gradient-based learning applied to document recognition. *Proceedings of the IEEE*, 86(11):2278–2324, 1998.

[12] J. Mutch and D. G. Lowe. Multiclass object recognition with sparse, localized features. *IEEE Conference on Computer Vision and Pattern Recognition*, 2006.

[13] P. J. Philips, P. J. Flynn, T. Scruggs, K. W. Bower, and W. Worek. Preliminary face recognition grand challenge results. *IEEE Conference on Automatic Face and Gesture Recgonition*, 2006.

[14] R. Raina, A. Battle, H. Lee, B. Packer, and A. Y. Ng. Self-taught learning: Transfer learning from unlabeled data. *International Conference on Machine Learning*, 2007.

[15] M. Ranzato, F.-J. Huang, Y.-L. Boureau, and Y. LeCun. Unsupervised learning of invariant feature hierarchies with applications to object recognition. *IEEE Conference on Computer Vision and Pattern Recognition*, 2007.

[16] M. Ranzato and M. Szummer. Semi-supervised learning of compact document representations with deep networks. *International Conferenece on Machine Learning*, 2008.

[17] T. Serre, L. Wolf, and T. Poggio. Object recognition with features inspired by visual cortex. *IEEE Conference on Computer Vision and Pattern Recognition*, 2005.

[18] J. Weston, F. Ratle, and R. Collobert. Deep learning via semi-supervised embedding. *International Conference on Machine Learning*, 2008.

[19] C. Williams and M. Seeger. Using the Nyström method to speed up kernel machines. *Neural Information Processing Systems*, 2001.

